# The Coloured Noise Expansion and Parameter Estimation of Diffusion Processes

**Simon M.J. Lyons**
School of Informatics
University of Edinburgh
10 Crichton Street, Edinburgh, EH8 9AB
S.Lyons-4@sms.ed.ac.uk

**Simo Särkkä**
Aalto University
Department of Biomedical Engineering
and Computational Science
Rakentajanaukio 2, 02150 Espoo
simo.sarkka@aalto.fi

**Amos J. Storkey**
School of Informatics
University of Edinburgh
10 Crichton Street, Edinburgh, EH8 9AB
a.storkey@ed.ac.uk

## Abstract

Stochastic differential equations (SDE) are a natural tool for modelling systems that are inherently noisy or contain uncertainties that can be modelled as stochastic processes. Crucial to the process of using SDE to build mathematical models is the ability to estimate parameters of those models from observed data. Over the past few decades, significant progress has been made on this problem, but we are still far from having a definitive solution. We describe a novel method of approximating a diffusion process that we show to be useful in Markov chain Monte-Carlo (MCMC) inference algorithms. We take the 'white' noise that drives a diffusion process and decompose it into two terms. The first is a 'coloured noise' term that can be deterministically controlled by a set of auxilliary variables. The second term is small and enables us to form a linear Gaussian 'small noise' approximation. The decomposition allows us to take a diffusion process of interest and cast it in a form that is amenable to sampling by MCMC methods. We explain why many state-of-the-art inference methods fail on highly nonlinear inference problems, and we demonstrate experimentally that our method performs well in such situations. Our results show that this method is a promising new tool for use in inference and parameter estimation problems.

## 1 Introduction

Diffusion processes are a flexible and useful tool in stochastic modelling. Many important real world systems are currently modelled and best understood in terms of stochastic differential equations in general and diffusions in particular. Diffusions have been used to model prices of financial instruments [1], chemical reactions [2], firing patterns of individual neurons [3], weather patterns [4] and fMRI data [5, 6, 7] among many other phenomena.

The analysis of diffusions dates back to Feller and Kolmogorov, who studied them as the scaling limits of certain Markov processes (see [8]). The theory of diffusion processes was revolutionised by Itô, who interpreted a diffusion process as the solution to a stochastic differential equation [9, 10]. This viewpoint allows one to see a diffusion process as the randomised counterpart of an ordinary differential equation. One can argue that stochastic differential equations are the natural

tool for modelling continuously evolving systems of real valued quantities that are subject to noise or stochastic influences.

The classical approach to mathematical modelling starts with a set of equations that describe the evolution of a system of interest. These equations are goverened by a set of input parameters (for example particle masses, reaction rates, or more general constants of proportionality) that determine the behaviour of the system. For practical purposes, it is of considerable interest to solve the *inverse problem*. Given the output of some system, what can be said about the parameters that govern it?

In the present setting, we observe data which we hypothesize are generated by a diffusion. We would like to know what the nature of this diffusion is. For example, we may begin with a parametric model of a physical system, with a prior distribution over the parameters. In principle, one can apply Bayes' theorem to deduce the posterior distribution. In practice, this is computationally prohibitive: it is necessary to solve a partial differential equation known as the Fokker-Planck equation (see [11]) in order to find the transition density of the diffusion of interest. This solution is rarely available in closed form, and must be computed numerically.

In this paper, we propose a novel approximation for a nonlinear diffusion process $\mathbf{X}$. One heuristic way of thinking about a diffusion is as an ordinary differential equation that is perturbed by white noise. We demonstrate that one can replace the white noise by a 'coloured' approximation without inducing much error. The nature of the coloured noise expansion method enables us to control the behaviour of the diffusion over various length-scales. This allows us to produce samples from the diffusion process that are consistent with observed data. We use these samples in a Markov chain Monte-Carlo (MCMC) inference algorithm.

The main contributions of this paper are:

- Novel development of a method for sampling from the time-$t$ marginal distribution of a diffusion process based on a 'coloured' approximation of white noise.
- Demonstration that this approximation is a powerful and scalable tool for making parameter estimation feasible for general diffusions at minimal cost.

The paper is structured as follows: in Section 2, we describe the structure of our problem. In Section 3 we conduct a brief survey of existing approaches to the problem. In Section 4, we discuss the coloured noise expansion and its use in controlling the behaviour of a diffusion process. Our inference algorithm is described in Section 5. We describe some numerical experiments in Section 6, and future work is discussed in Section 7.

## 2 Parametric Diffusion Processes

In this section we develop the basic notation and formalism for the diffusion processes used in this work. First, we assume our data are generated by observing a $k$-dimensional diffusion processes with dynamics

$$d\mathbf{X}_t = \mathbf{a}_\theta(\mathbf{X}_t)dt + \mathbf{B}_\theta d\mathbf{W}_t, \qquad \mathbf{X}_0 \sim p(\mathbf{x}_0), \tag{1}$$

where the initial condition is drawn from some known distribution. Observations are assumed to occur at times $t_1, \ldots, t_n$, with $t_i - t_{i-1} := T_i$. We require that $a_\theta : I\!\!R^k \to I\!\!R^k$ is sufficiently regular to guarantee the existence of a unique strong solution to (1), and we assume $\mathbf{B}_\theta \in I\!\!R^{k \times d}$. Both terms depend on a set of potentially unknown parameters $\theta \in I\!\!R^{d_\theta}$. We impose a prior distribution $p(\theta)$ on the parameters. The driving noise $\mathbf{W}$ is a $d$-dimensional Brownian motion, and the equation is interpreted in the Itô sense. Observations are subject to independent Gaussian perturbations centered at the true value of $\mathbf{X}$. That is,

$$\mathbf{Y}_{t_i} = \mathbf{X}_{t_i} + \epsilon_{t_i}, \qquad \epsilon_{t_i} \sim \mathcal{N}(0, \Sigma_i) \tag{2}$$

We use the notation $\mathbf{X}$ to refer to the entire sample path of the diffusion, and $\mathbf{X}_t$ to denote the value of the process at time $t$. We will also employ the shorthand $\mathbf{Y}_{1:n} = \{\mathbf{Y}_{t_1}, \ldots, \mathbf{Y}_{t_n}\}$.

Many systems can be modelled using the form (1). Such systems are particularly relevant in physics and natural sciences. In situations where this is not explicitly the case, one can often hope to reduce a diffusion to this form via the Lamperti transform. One can almost always accomplish this in the uni-variate case, but the multivariate setting is somewhat more involved. Aït-Sahalia [12] characterises the set of multivariate diffusions to which this transform can be applied.

## 3 Background Work

Most approaches to parameter estimation of diffusion processes rely on the Monte-Carlo approximation. Beskos et al. [13] [14] employ a method based on rejection sampling to estimate parameters without introducing any discretisation error. Golightly and Wilkinson [15] extend the work of Chib et al. [16] and Durham and Gallant [17] to construct a Gibbs sampler that can be applied to the parameter estimation problem.

Roughly speaking, Gibbs samplers that exist in the literature alternate between drawing samples from some representation of the diffusion process $\mathbf{X}$ conditional on parameters $\theta$, and samples from $\theta$ conditional on the current sample path of $\mathbf{X}$. Note that draws from $\mathbf{X}$ must be consistent with the observations $\mathbf{Y}_{1:n}$.

The usual approach to the consistency issue is to make a proposal by conditioning a linear diffusion to hit some neighbourhood of the observation $\mathbf{Y}_k$, then to make a correction via a rejection sampling [18] or a Metropolis-Hastings [16] step. However, as the inter-observation time grows, the qualitative difference between linear and nonlinear diffusions gets progressively more pronounced, and the rate of rejection grows accordingly. Figure 1 shows the disparity between a sample from a nonlinear process and a sample from the linear proposal. One can see that the target sample path is constrained to stay near the mode $\gamma = 2.5$, whereas the proposal can move more freely. One should expect to make many proposals before finding one that 'behaves' like a typical draw from the true process.

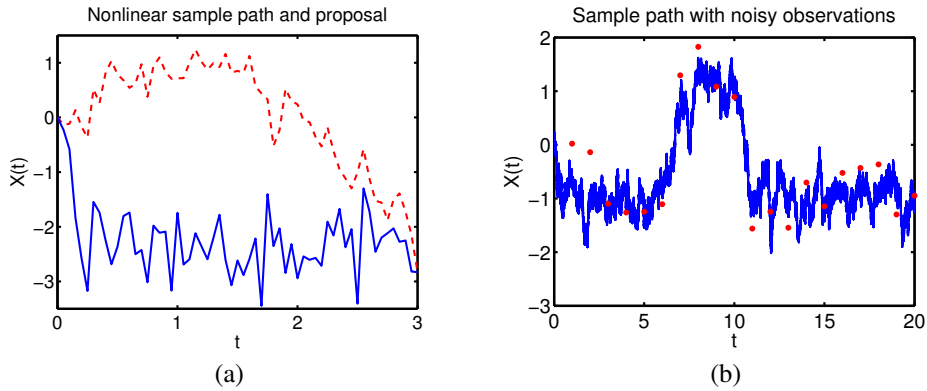

Figure 1: (a) Sample path of a double well process (see equation (18)) with $\alpha = 2$, $\gamma = 2.5$, $B = 2$ (blue line). Current Gibbs samplers use linear proposals (dashed red line) with a rejection step to draw conditioned nonlinear paths. In this case, the behaviour of the proposal is very different to that of the target, and the rate of rejection is high.
(b) Sample path of a double well process (solid blue line) with noisy observations (red dots). We use this as an initial dataset on which to test our algorithm. Parameters are $\alpha = 2, \gamma = 1, B = 1$. Observation errors have variance $\Sigma = .25$.

For low-dimensional inference problems, algorithms that employ sequential Monte-Carlo (SMC) methods [19] [20] typically yield good results. However, unlike the Gibbs samplers mentioned above, SMC-based methods often do not scale well with dimension. The number of particles that one needs to maintain a given accuracy is known to scale exponentially with the dimension of the problem [21].

Aït-Sahalia [12, 22] uses a deterministic technique based on Edgeworth expansions to approximate the transition density. Other approaches include variational methods [23, 24] that can compute continuous time Gaussian process approximations to more general stochastic differential systems, as well as various non-linear Kalman filtering and smoothing based approximations [25, 26, 27] .

## 4 Coloured Noise Expansions and Brownian Motion

We now introduce a method of approximating a nonlinear diffusion that allows us to gain a considerable amount of control over the behaviour of the process. Similar methods have been used

for stratified sampling of diffusion processes [28] and the solution of stochastic partial differential equations [29] . One of the major challenges of using MCMC methods for parameter estimation in the present context is that it is typically very difficult to draw samples from a diffusion process conditional on observed data. If one only knows the initial condition of a diffusion, then it is straight-forward to simulate a sample path of the process. However, simulating a sample path conditional on both initial and *final* conditions is a challenging problem.

Our approximation separates the diffusion process $\mathbf{X}$ into the sum of a linear and nonlinear component. The linear component of the sum allows us to condition the approximation to fit observed data more easily than in conventional methods. On the other hand, the nonlinear component captures the 'gross' variation of a typical sample path. In this section, we fix a generic time interval $[0, T]$, though one can apply the same derivation for any given interval $T_i = t_i - t_{i-1}$.

Heuristically, one can think of the random process that drives the process defined in equation (1) as white noise. In our approximation, we project this white noise into an $N$-dimensional subspace of $L^2[0, T]$, the Hilbert space of square-integrable functions defined on the interval $[0, T]$. This gives a 'coloured noise' process that approaches white noise asymptotically as $N \to \infty$. The coloured noise process is then used to drive an approximation of (1). We can choose the space into which to project the white noise in such a way that we will gain some control over its behaviour. This is analogous to the way that Fourier analysis allows us to manipulate properties of signals

Recall that a standard Brownian motion on the interval $[0, T]$ is a one-dimentional Gaussian process with zero mean and covariance function $k(s, t) = \min\{s, t\}$. By definition of the Itô integral, we can write

$$
W_t = \int_0^t dW_s = \int_0^T \mathbb{I}_{[0,t]}(s) dW_s. \tag{3}
$$

Suppose $\{\phi_i\}_{i \geq 1}$ is an orthonormal basis of $L^2[0, T]$. We can interpret the indicator function in (3) as an element of $L^2[0, T]$ and expand it in terms of the basis functions as follows:

$$
\mathbb{I}_{[0,t]}(s) = \sum_{i=1}^{\infty} \langle \mathbb{I}_{[0,t]}(\cdot), \phi_i(\cdot) \rangle \phi_i(s) = \sum_{i=1}^{\infty} \left( \int_0^t \phi_i(u) du \right) \phi_i(s). \tag{4}
$$

Substituting (4) into (3), we see that

$$
W_t = \sum_{i=1}^{\infty} \left( \int_0^T \phi_i(s) dW_s \right) \int_0^t \phi_i(u) du. \tag{5}
$$

We will employ the shorthand $Z_i = \int_0^T \phi_i(s) dW_s$. Since the functions $\{\phi_i\}$ are deterministic and orthonormal, we know from standard results of Itô calculus that the random variables $\{Z_i\}$ are i.i.d standard normal.

The infinite series in equation (5) can be truncated after $N$ terms to derive an approximation, $\hat{W}_t$ of Brownian motion. Taking the derivative with respect to time, the result is a 'coloured' approximation of white noise, taking the form

$$
\frac{d\hat{W}_t}{dt} = \sum_{i=1}^{N} Z_i \phi_i(t). \tag{6}
$$

The multivariate approximation is similar. We seperate a $d$-dimensional Brownian motion into one-dimensional components and decompose the individual components as in (6). In principle, one can choose a different value of $N$ for each component of the Brownian motion, but for ease of exposition we do not do so here. We can substitute this approximation into equation (1), which gives

$$
\frac{d\mathbf{X}_t^{\mathrm{NL}}}{dt} = \mathbf{a}_\theta(\mathbf{X}_t^{\mathrm{NL}}) + \mathbf{B}_\theta \sum_{i=1}^{N} \mathbf{\Phi}_i \mathbf{Z}_i, \qquad \mathbf{X}_0^{\mathrm{NL}} \sim p(\mathbf{x}_0), \tag{7}
$$

where $\mathbf{\Phi}_i$ is the diagonal $d \times d$ matrix with entries $(\phi_{i1}, \ldots, \phi_{id})$, and $\mathbf{Z}_i = (Z_{i1}, \ldots, Z_{id})^\mathsf{T}$.

This derivation is useful because equation (7) gives us an alternative to the Euler-Maruyama discretisation for sampling approximately from the time-$t$ marginal distribution of a diffusion process. We

draw coefficients $Z_{ij}$ from a standard normal distribution, and solve the appropriate vector-valued *ordinary* differential equation. While the Euler discretisation is the de facto standard method for numerical approximation of SDE, other methods do exist. Kloeden and Platen [30] discuss higher order methods such as the stochastic Runge-Kutta scheme [31].

In the Euler-Maruyama approximation, one discretises the driving Brownian motion into increments $W_{t_i} - W_{t_{i-1}} = \sqrt{T_i} Z_i$. One must typically employ a fine discretisation to get a good approximation to the true diffusion process. Empirically, we find that one needs far fewer Gaussian inputs $Z_i$ for an accurate representation of $\mathbf{X}_T$ using the coloured noise approximation. This more parsimonious representation has advantages. For example, Corlay and Pages [28] employ related ideas to conduct stratified sampling of a diffusion process.

The coefficients $\mathbf{Z}_i$ are also more amenable to interpretation than the Gaussian increments in the Euler-Maruyama expansion. Suppose we have a one-dimensional process in which we use the Fourier cosine basis

$$\phi_k(t) = \sqrt{2/T} \cos((2k-1)\pi t/2T). \tag{8}$$

If we change $Z_1$ while holding the other coefficients fixed, we will typically see a change in the large-scale behaviour of the path. On the other hand, a change in $Z_N$ will typically result in a change to the small-scale oscillations in the path. The seperation of behaviours across coefficients gives us a means to obtain fine-grained control over the behaviour of a diffusion process within a Metropolis-Hastings algorithm.

We can improve our approximation by attempting to correct for the fact that we truncated the sum in equation (6). Instead of simply discarding the terms $Z_i \Phi_i$ for $i > N$, we attempt to account for their effect as follows. We assume the existence of some 'correction' process $\mathbf{X}^C$ such that $\mathbf{X} = \mathbf{X}^{NL} + \mathbf{X}^C$. We know that the dynamics of $\mathbf{X}$ satisfy

$$d\mathbf{X}_t = \mathbf{a}_\theta \left( \mathbf{X}_t^{NL} + \mathbf{X}_t^C \right) dt + \mathbf{B}_\theta d\mathbf{W}_t. \tag{9}$$

Taylor expanding the drift term around $\mathbf{X}^{NL}$, we see that to first order,

$$d\mathbf{X}_t \approx \left( \mathbf{a}_\theta \left( \mathbf{X}_t^{NL} \right) + \mathbf{J}_\mathbf{a}(\mathbf{X}_t^{NL}) \mathbf{X}_t^C \right) dt + \mathbf{B}_\theta d\mathbf{W}_t$$

$$= \left( \mathbf{a}_\theta \left( \mathbf{X}_t^{NL} \right) + \mathbf{B}_\theta d\hat{\mathbf{W}}_t \right) dt + \mathbf{J}_\mathbf{a}(\mathbf{X}_t^{NL}) \mathbf{X}_t^C dt + \mathbf{B}_\theta \left( d\mathbf{W}_t - d\hat{\mathbf{W}}_t \right). \tag{10}$$

Here, $\mathbf{J}_\mathbf{a}(x)$ is the Jacobian matrix of the function $a$ evaluated at $x$. This motivates the use of a linear time-dependent approximation to the correction process. We will refer to this linear approximation as $\mathbf{X}^L$. The dynamics of $\mathbf{X}^L$ satisfy

$$d\mathbf{X}_t^L = \mathbf{J}_\mathbf{a}(\mathbf{X}_t^{NL}) \mathbf{X}_t^L dt + \mathbf{B}_\theta d\mathbf{R}_t, \qquad \mathbf{X}_0^L = 0, \tag{11}$$

where the driving noise is the 'residual' term $\mathbf{R} = \mathbf{W} - \hat{\mathbf{W}}$. Conditional on $\mathbf{X}^{NL}$, $\mathbf{X}^L$ is a linear Gaussian process, and equation (11) can be solved in semi-closed form. First, we compute a numerical approximation to the solution of the homogenous matrix-valued equation

$$\frac{d}{dt} \Psi(t) = \mathbf{J}_\mathbf{a}(\mathbf{X}_t^{NL}) \Psi(t), \qquad \Psi(0) = \mathbf{I}_n. \tag{12}$$

One can compute $\Psi^{-1}(t)$ in a similar fashion via the relationship $d\Psi^{-1}/dt = -\Psi^{-1}(d\Psi/dt)\Psi^{-1}$.

We then have

$$\mathbf{X}_t^L = \Psi(t) \int_0^t \Psi(u)^{-1} \mathbf{B} d\mathbf{R}_u$$

$$= \Psi(t) \int_0^t \Psi(u)^{-1} \mathbf{B} d\mathbf{W}_u - \sum_{i=1}^N \Psi(t) \left( \int_0^t \Psi(u)^{-1} \mathbf{B} \mathbf{\Phi}_i(u) du \right) \mathbf{Z}_i. \tag{13}$$

It follows that $\mathbf{X}^{\mathrm{L}}$ has mean 0 and covariance

$$
\begin{aligned}
k(s,t) = {} & \Psi(s) \left( \int_0^{s \wedge t} \Psi(u)^{-1} \mathbf{B} \mathbf{B}^{\mathsf{T}} \Psi^{\mathsf{T}}(u)^{-1} du \right) \Psi^{\mathsf{T}}(t) \\
& - \sum_{i=1}^N \Psi(s) \left( \int_0^s \Psi(u)^{-1} \mathbf{B} \boldsymbol{\Phi}_i(u) du \right) \left( \int_0^t \Psi(u)^{-1} \mathbf{B} \boldsymbol{\Phi}_i(u) du \right)^{\mathsf{T}} \Psi^{\mathsf{T}}(t). \quad (14)
\end{aligned}
$$

The process $\mathbf{X}^{\mathrm{NL}}$ is designed to capture the most significant nonlinear features of the original diffusion $\mathbf{X}$, while the linear process $\mathbf{X}^{\mathrm{L}}$ corrects for the truncation of the sum (6), and can be understood using tools from the theory of Gaussian processes. One can think of the linear term as the result of a 'small-noise' expansion *about the nonlinear trajectory*. Small-noise techniques have been applied to diffusions in the past [11], but the method described above has the advantage of being inherently nonlinear. In the supplement to this paper, we show that $\hat{\mathbf{X}} = \mathbf{X}^{\mathrm{NL}} + \mathbf{X}^{\mathrm{L}}$ converges to $\mathbf{X}$ in $L^2[0,T]$ as $N \to \infty$ under the assumption that $\mathbf{a}$ is Lipschitz continuous. If the drift function is linear, then $\hat{\mathbf{X}} = \mathbf{X}$ regardless of the choice of $N$.

## 5   Parameter Estimation

In this section, we describe a novel modification of the Gibbs sampler that does not suffer the drawbacks of the linear proposal strategy. In Section 6, we demonstrate that for highly nonlinear problems it will perform significantly better than standard methods because of the nonlinear component of our approximation.

Suppose for now that we make a single noiseless observation at time $t_1 = T$ (for ease of notation, we will assume that observations are uniformly spaced through time with $t_{i+1} - t_i = T$, though this is not necessary). Our aim is to sample from the posterior distribution

$$
p\left(\theta, \mathbf{Z}_{1:N} \,\middle|\, \mathbf{X}_1^{\mathrm{NL}} + \mathbf{X}_1^{\mathrm{L}} = \mathbf{Y}_1\right) \propto \mathcal{N}(\mathbf{Y}_1 \mid \mathbf{X}_1^{\mathrm{NL}}, k_1(T,T)) \mathcal{N}(\mathbf{Z}_{1:N}) p(\theta). \quad (15)
$$

We adopt the convention that $\mathcal{N}(\cdot \mid \mu, \Sigma)$ represents the normal distribution with mean $\mu$ and covariance $\Sigma$, whereas $\mathcal{N}(\cdot)$ represents the standard normal distribution. Note that we have left dependence of $k_1$ on $\mathbf{Z}$ and $\theta$ implicit. The right-hand side of this expression allows us to evaluate the posterior up to proportionality; hence it can be targeted with a Metropolis-Hastings sampler.

With multiple observations, the situation is similar. However, we now have a set of Gaussian inputs $\mathbf{Z}^{(i)}$ for each transition $\hat{\mathbf{X}}_i | \hat{\mathbf{X}}_{i-1}$. If we attempt to update $\theta$ and $\{\mathbf{Z}^{(i)}\}_{i \leq n}$ all at once, the rate of rejection will be unacceptably high. For this reason, we update each $\mathbf{Z}^{(i)}$ in turn, holding $\theta$ and the other Gaussian inputs fixed. We draw $\mathbf{Z}^{(i)*}$ from the proposal distribution, and compute $\mathbf{X}_i^{\mathrm{NL}*}$ with initial condition $Y_{i-1}$. We also compute the covariance $k_i^*(T,T)$ of the linear correction. The acceptance probability for this update is

$$
\alpha = 1 \wedge \frac{\mathcal{N}(\mathbf{Y}_i \mid \mathbf{X}_i^{\mathrm{NL}*}, k_i^*(T,T)) \mathcal{N}(\mathbf{Z}_{1:N}^{(i)*}) p(\mathbf{Z}_{1:N}^{(i)*} \to \mathbf{Z}_{1:N}^{(i)})}{\mathcal{N}(\mathbf{Y}_i \mid \mathbf{X}_i^{\mathrm{NL}}, k_i(T,T)) \mathcal{N}(\mathbf{Z}_{1:N}^{(i)}) p(\mathbf{Z}_{1:N}^{(i)} \to \mathbf{Z}_{1:N}^{(i)*})} \quad (16)
$$

After updating the Gaussian inputs, we make a global update for the $\theta$ parameter. The acceptance probability for this move is

$$
\alpha = 1 \wedge \prod_{i=1}^n \frac{\mathcal{N}(\mathbf{Y}_i \mid \mathbf{X}_i^{\mathrm{NL}*}, k_i^*(T,T)) p(\theta^*) p(\theta^* \to \theta)}{\mathcal{N}(\mathbf{Y}_i \mid \mathbf{X}_i^{\mathrm{NL}}, k_i(T,T)) p(\theta) p(\theta \to \theta^*)}, \quad (17)
$$

where $\mathbf{X}_i^{\mathrm{NL}*}$ and $k_i^*(T,T)$ are computed using the proposed value of $\theta^*$.

We noted earlier that when $j$ is large, $\mathbf{Z}_j$ governs the small-time oscillations of the diffusion process. One should not expect to gain much information about the value of $\mathbf{Z}_j$ when we have large interobservation times. We find this to be the case in our experiments - the posterior distribution of $\mathbf{Z}_{j:N}$ approaches a spherical Gaussian distribution when $j > 3$. For this reason, we employ a Gaussian random walk proposal in $\mathbf{Z}_1$ with stepsize $\sigma_{\mathrm{RW}} = .45$, and proposals for $\mathbf{Z}_{2:N}$ are drawn independently from the standard normal distribution.

In the presence of observation noise, we proceed roughly as before. Recall that we make observations $\mathbf{Y}_i = \mathbf{X}_i + \epsilon_i$. We draw proposals $\mathbf{Z}_{1:N}^{(i)*}$ and $\epsilon_i^*$. The initial condition for $\mathbf{X}_i^{\mathrm{NL}}$ is now $\mathbf{Y}_{i-1} - \epsilon_{i-1}$. However, one must make an important modification to the algorithm. Suppose we propose an update of $\hat{\mathbf{X}}_i$ and it is accepted. If we subsequently propose an update for $\hat{\mathbf{X}}_{i+1}$ and it is rejected, then the initial condition for $\hat{\mathbf{X}}_{i+1}$ will be inconsistent with the current state of the chain (it will be $\mathbf{Y}_i - \epsilon_i$ instead of $\mathbf{Y}_i - \epsilon_i^*$). For this reason, we must propose joint updates for $(\hat{\mathbf{X}}_i, \epsilon_i, \hat{\mathbf{X}}_{i+1})$. If the variance of the observation noise is high, it may be more efficient to target the joint posterior distribution $p\left(\theta, \{\mathbf{Z}_{1:N}^i, \mathbf{X}_i^{\mathrm{L}}\} \mid \mathbf{Y}_{1:n}\right)$.

## 6 Numerical Experiments

The double-well diffusion is a widely-used benchmark for nonlinear inference problems [24, 32, 33, 34]. It has been used to model systems that exhibit switching behaviour or bistability [11, 35]. It possesses nonlinear features that are sufficient to demonstrate the shortcomings of some existing inference methods, and how our approach overcomes these issues. The dynamics of the process are given by

$$dX_t = \alpha X_t \left(\gamma^2 - X_t^2\right) dt + B dW_t. \tag{18}$$

The process $X$ has a bimodal stationary distribution, with modes at $x = \pm\gamma$. The parameter $\alpha$ governs the rate at which sample trajectories are 'pushed' toward either mode. If $B$ is small in comparison to $\alpha$, mode-switching occurs relatively rarely.

Figure 1(b) shows a trajectory of a double-well diffusion over 20 units of time, with observations at times $\{1, 2, \ldots, 20\}$ . We used the parameters $\alpha = 2$, $\gamma = 1$, $B = 1$. The variance of the observation noise was set to $\Sigma = .25$.

As we mentioned earlier, particle MCMC performs well in low-dimensional inference problems. For this reason, the results of a particle MCMC inference algorithm (with $N = 1,000$) particles are used as 'ground truth'. Our algorithm used $N = 3$ Gaussian inputs with a linear correction. We used the Fourier cosine series (8) as an orthonormal basis. We compare our Gibbs sampler to that of Golightly and Wilkinson [15], for which we use an Euler discretisation with stepsize $\Delta t = .05$. Each algorithm drew $70,000$ samples from the posterior distribution, moving through the parameter space in a Gaussian random walk. We placed an exponential(4) prior on $\gamma$ and an exponential(1) prior on $\alpha$ and $B$.

For this particular choice of parameters, both Gibbs samplers give a good approximation to the true posterior. Figure 2 shows histograms of the marginal posterior distributions of $(\alpha, \gamma, B)$ for each algorithm.

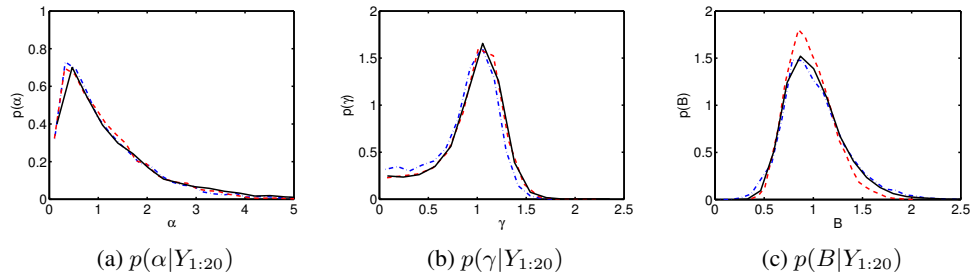

    (a) $p(\alpha|Y_{1:20})$           (b) $p(\gamma|Y_{1:20})$           (c) $p(B|Y_{1:20})$

Figure 2: Marginal posterior distributions for $(\alpha, \gamma, B)$ conditional on observed data. The solid black line is the output of a particle MCMC method, taken as ground truth. The broken red line is the output of the linear proposal method, and the broken and dotted blue line is the density estimate from the coloured noise expansion method. We see that both methods give a good approximation to the ground truth.

Gibbs samplers that have been used in the past rely on making proposals by conditioning a linear diffusion to hit a target, and subsequently accepting or rejecting those proposals. Over short timescales, or for problems that are not highly nonlinear, this can be an effective strategy. However, as the timescale increases, the proposal and target become quite dissimilar (see Figure 1(a)).

For our second experiment, we simulate a double well process with $(\alpha, \gamma, B) = (2, 2.5, 2)$. We make noisy observations with $t_i - t_{i-1} = 3$ and $\Sigma = .1$. The algorithms target the posterior distribution over $\gamma$, with $\alpha$ and $B$ fixed at their true values. From our previous discussion, one might expect the linear proposal strategy to perform poorly in this more nonlinear setting. This is indeed the case. As in the previous experiment, we used a linear proposal Gibbs sampler with Euler stepsize $dt = 0.05$. In the 'path update' stage, fewer than .01% of proposals were accepted. On the other hand, the coloured noise expansion method used $N = 7$ Gaussian inputs with a linear correction and was able to approximate the posterior accurately. Figure 3 shows histograms of the results. Note the different scaling of the rightmost plot.

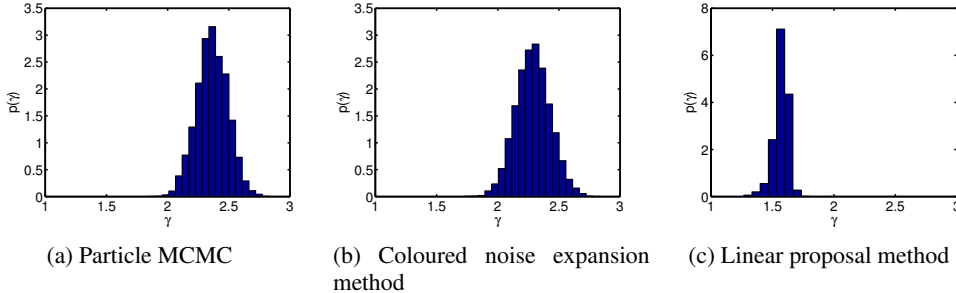

(a) Particle MCMC      (b) Coloured noise expansion method      (c) Linear proposal method

Figure 3: $p(\gamma|Y_{1:10}, B, \alpha)$ after ten observations with a relatively large inter-observation time. We drew data from a double well process with $(\alpha, \gamma, B) = (2, 2.5, 2)$. The coloured noise expansion method matches the ground truth, whereas the linear proposal method is inconsistent with the data.

# 7 Discussion and Future Work

We have seen that the standard linear proposal/correction strategy can fail for highly nonlinear problems. Our inference method avoids the linear correction step, instead targeting the posterior over input variables *directly*. With regard to computational efficiency, it is difficult to give an authoritative analysis because both our method and the linear proposal method are complex, with several parameters to tune. In our experiments, the algorithms terminated in a roughly similar length of time (though no serious attempt was made to optimise the runtime of either method).

With regard to our method, several questions remain open. The accuracy of our algorithm depends on the choice of basis functions $\{\phi_i\}$. At present, it is not clear how to make this choice optimally in the general setting. In the linear case, it is possible to show that one can achieve the accuracy of the Karhunen-Loeve decomposition, which is theoretically optimal. One can also set the error at a single time $t$ to zero with a judicious choice of a single basis function. We aim to present these results in a paper that is currently under preparation.

We used a Taylor expansion to compute the covariance of the correction term. However, it may be fruitful to use more sophisticated ideas, collectively known as statistical linearisation methods. In this paper, we restricted our attention to processes with a state-independent diffusion coefficient so that the covariance of the correction term could be computed. We may be able to extend this methodology to process with state-dependent noise - certainly one could achieve this by taking a 0-th order Taylor expansion about $\mathbf{X}^{\mathrm{NL}}$. Whether it is possible to improve upon this idea is a matter for further investigation.

**Acknowledgments**

Simon Lyons was supported by Microsoft Research, Cambridge.

# References

[1] R.C. Merton. Theory of rational option pricing. *The Bell Journal of Economics and Management Science*, 4:141–183, 1973.

[2] D.T. Gillespie. The chemical Langevin equation. *Journal of Chemical Physics*, 113,1:297–306, 2000.

[3] G. Kallianpur. Weak convergence of stochastic neuronal models. *Stochastic Methods in Biology*, 70:116–145, 1987.

[4] H.A. Dijkstra, L.M. Frankcombe, and A.S. von der Heydt. A stochastic dynamical systems view of the Atlantic Multidecadal Oscillation. *Philosophical Transactions of the Royal Society A*, 366:2543–2558, 2008.

[5] L. Murray and A. Storkey. Continuous time particle filtering for fMRI. *Advances in Neural Information Processing Systems*, 20:1049–1056, 2008.

[6] J. Daunizeau, K.J. Friston, and S.J. Kiebel. Variational Bayesian identification and prediction of stochastic nonlinear dynamic causal models. *Physica D*, pages 2089–2118, 2009.

[7] L.M. Murray and A.J. Storkey. Particle smoothing in continuous time: A fast approach via density estimation. *IEEE Transactions on Signal Processing*, 59:1017–1026, 2011.

[8] W. Feller. *An Introduction to Probability Theory and its Applications, Volume II*. Wiley, 1971.

[9] I. Karatzas and S.E. Shreve. *Brownian Motion and Stochastic Calculus*. Springer, 1991.

[10] B. Oksendal. *Stochastic Differential Equations*. Springer, 2007.

[11] C.W. Gardiner. *Handbook of Stochastic Methods for Physics, Chemistry and the Natural Sciences*. Springer-Verlag, 1983.

[12] Y. Aït-Sahalia. Closed-form likelihood expansions for multivariate diffusions. *The Annals of Statistics*, 36(2):906–937, 2008.

[13] A. Beskos, O. Papaspiliopoulos, and G.O. Roberts. Monte-Carlo maximum likelihood estimation for discretely observed diffusion processes. *Annals of Statistics*, 37:223–245, 2009.

[14] A. Beskos, O. Papaspiliopoulos, G.O. Roberts, and P. Fearnhead. Exact and computationally efficient likelihood-based estimation for discretely observed diffusion processes (with discussion). *Journal of the Royal Statistical Society: Series B (Statistical Methodology)*, 68:333–382, 2006.

[15] A. Golightly and D.J. Wilkinson. Bayesian inference for nonlinear multivariate diffusion models observed with error. *CSDA*, 52:1674–1693, 2008.

[16] S. Chib, M.K. Pitt, and N. Shepard. Likelihood-based inference for diffusion models. *Working Paper*, 2004. http://www.nuff.ox.ac.uk/economics/papers/2004/w20/chibpittshephard.pdf.

[17] G.B. Durham and A.R. Gallant. Numerical techniques for maximum likelihood estimation of continuous-time diffusion processes (with comments). *Journal of Business and Economic Statistics*, 20:297–338, 2002.

[18] A. Beskos, O. Papaspiliopoulos, and G.O. Roberts. Retrospective exact simulation of diffusion sample paths with applications. *Bernoulli*, 12(6):1077, 2006.

[19] D. Rimmer, A. Doucet, and W.J. Fitzgerald. Particle filters for stochastic differential equations of nonlinear diffusions. Technical report, Cambridge University Engineering Department, 2005.

[20] Christophe Andrieu, Arnaud Doucet, and Roman Holenstein. Particle Markov Chain Monte Carlo methods. *Journal of the Royal Statistical Society*, 72:1–33, 2010.

[21] C. Snyder, T. Bengtsson, P. Bickel, and J. Anderson. Obstacles to high-dimensional particle filtering. *Monthly Weather Review*, 136(12):4629–4640, 2008.

[22] Y. Aït-Sahalia. Maximum likelihood estimation of discretely sampled diffusions: a closed-form approximation approach. *Econometrica*, 70:223–262, 2002.

[23] C. Archambeau, D. Cornford, M. Opper, and J. Shawe-Taylor. Gaussian process approximations of stochastic differential equations. *JMLR: Workshop and Conference Proceedings*, 1:1–16, 2007.

[24] C. Archambeau, M. Opper, Y. Shen, D. Cornford, and J. Shawe-Taylor. Variational inference for diffusion processes. In *Advances in Neural Information Processing Systems 20 (NIPS 2007)*, 2008.

[25] S. Särkkä. On unscented Kalman filtering for state estimation of continuous-time nonlinear systems. *IEEE Transactions on Automatic Control*, 52:1631–1641, 2007.

[26] A.H. Jazwinski. *Stochastic processes and filtering theory*, volume 63. Academic Pr, 1970.

[27] H. Singer. Nonlinear continuous time modeling approaches in panel research. *Statistica Neerlandica*, 62(1):29–57, 2008.

[28] S. Corlay and P. Gilles. Functional quantization based stratified sampling methods. *Arxiv preprint Arxiv:1008.4441*, 2010.

[29] W. Luo. *Wiener chaos expansion and numerical solutions of stochastic partial differential equations*. PhD thesis, California Institute of Technology, 2006.

[30] P.E. Kloeden and E. Platen. *Numerical Solution of Stochastic Differential Equations*. Springer, 1999.

[31] A.F. Bastani and S.M. Hosseini. A new adaptive Runge-Kutta method for stochastic differential equations. *Journal of Computational and Applied Mathematics*, 206:631–644, 2007.

[32] Y. Shen, C. Archambeau, D. Cornford, M. Opper, J. Shawe-Taylor, and R. Barillec. A comparison of variational and Markov chain Monte Carlo methods for inference in partially observed stochastic dynamic systems. *Journal of Signal Processing Systems*, 61(1):51–59, 2010.

[33] H. Singer. Parameter estimation of nonlinear stochastic differential equations: simulated maximum likelihood versus extended Kalman filter and Itô-Taylor expansion. *Journal of Computational and Graphical Statistics*, 11(4):972–995, 2002.

[34] M. Opper, A. Ruttor, and G. Sanguinetti. Approximate inference in continuous time Gaussian-jump processes. *Advances in Neural Information Processing Systems*, 23:1831–1839, 2010.

[35] N.G. van Kampen. *Stochastic processes in physics and chemistry*. North holland, 2007.

